# CODA: A Correlation-Oriented Disentanglement and Augmentation Modeling Scheme for Better Resisting Subpopulation Shifts

**Ziquan OU**
Department of Data Science
City University of Hong Kong
ziquanou2-c@my.cityu.edu.hk

**Zijun ZHANG**[*]
Department of Data Science
City University of Hong Kong
zijzhang@cityu.edu.hk

## Abstract

Data-driven models learned often struggle to generalize due to widespread subpopulation shifts, especially the presence of both spurious correlations and group imbalance (SC-GI). To learn models more powerful for defending against SC-GI, we propose a **Correlation-Oriented Disentanglement and Augmentation (CODA)** modeling scheme, which includes two unique developments: (1) correlation-oriented disentanglement and (2) strategic sample augmentation with reweighted consistency (RWC) loss. In (1), a bi-branch encoding process is developed to enable the disentangling of variant and invariant correlations by coordinating with a decoy classifier and the decoder reconstruction. In (2), a strategic sample augmentation based on disentangled latent features with RWC loss is designed to reinforce the training of a more generalizable model. The effectiveness of CODA is verified by benchmarking against a set of SOTA models in terms of worst-group accuracy and maximum group accuracy gap based on two famous datasets, ColoredMNIST and CelebA.

## 1 Introduction

One grand challenge that impedes the generalization of machine learning models is 'Subpopulation shifts', the alterations in the relative proportions of subpopulations between training and testing datasets [19]. In reality, one frequently encountered scenario is that specific groups in data are underrepresented, which we tag as the *group imbalance (GI)*, contributed by class imbalance (CI), attribute imbalance (AI), or their combination [29, 15]. Due to GI, models trained to minimize average loss tend to favor the majority groups and thus fail to generalize across minority groups [25].

Another widely discussed issue in subpopulation shifts is the *spurious correlations (SC)*, the incidental correlations between non-causal features and labels during training time that do not hold in the real world. SC can significantly compromise the efficacy of a model [1] and has been observed in various applications, such as speech recognition [10], medical imaging [27], object recognition [25, 8], image captioning [7], etc. For instance, object recognition models that classify animal species have been shown to generalize poorly to new environments since their classifications mistakenly favor image backgrounds over biological characteristics of animals [2, 28]. Learning such correlations can result in degraded performance on data breaking these spurious patterns, e.g., when animals are moved indoors or to an uncommon context, such as cows on a beach instead of in a pasture.

Prior research has extensively explored the developments of robust models performing uniformly well across subpopulations. Reported studies have varied from attempts of developing models without

---

[*]Correspondence Author

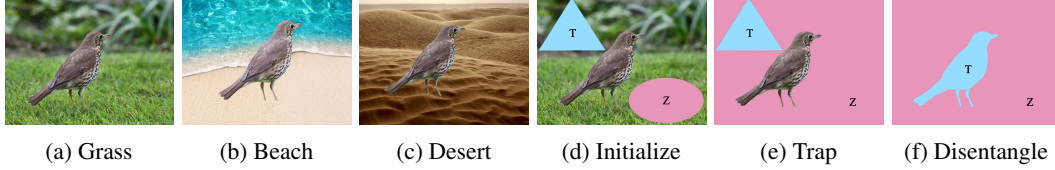

| (a) Grass | (b) Beach | (c) Desert | (d) Initialize | (e) Trap | (f) Disentangle |

Figure 1: (a) ~ (c): Same bird embedded in different backgrounds. A robust model is expected to predict consistently well on objects in all images. (d) ~ (f): The feature extraction and exchange process of COD. Z represents spurious correlations, and T represents causal correlations.

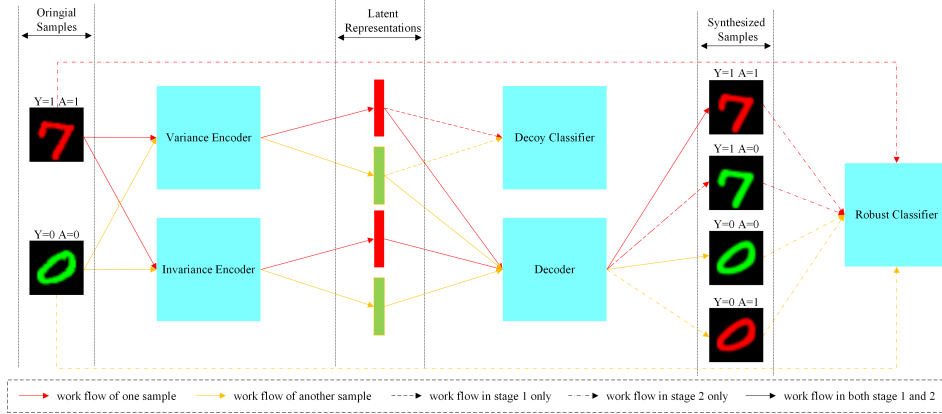

Figure 2: Overview of the CODA Framework on ColoredMNIST dataset. In ColoredMNIST, samples with digit less than 5 are negative samples ($Y = 0$), and the rest are positive samples ($Y = 1$). Each sample is either painted red ($A = 1$) or green ($A = 0$), with color spuriously correlated with the label in training set. In stage one, CODA first learns to encode and disentangle causal correlations from spurious ones. The decoder reconstructs the input data from the two encodings, while the decoy classifier is employed to lure the spurious information flowing towards the variance encoder. In stage two, CODA further enhances robustness by creating synthesized samples through the recombination of encoded features from different inputs, ensuring that the resultant samples maintain original class information but vary in spurious attributes. Finally, a robust classifier is trained with a novel reweighted consistency loss to deliver consistent performance across both original and synthetic samples, thereby reinforcing its resilience to spurious correlations.

pre-identified spurious attributes [21, 23, 26, 20] to methods leveraging knowledge of these spurious attributes to inform and guide training processes [25, 13, 30, 14, 15]. A critical observation in [21] unveiled that methods trained without utilizing spurious attributes still rely on them in the model selection; otherwise, a significant performance drop would be observed. The potential of better utilizing spurious attributes to further strengthen models in defending subpopulation shifts has yet to be fully explored, especially in terms of more innovations in utilizing spurious attributes model selection, addressing SC by mixing, or tackling GI via reweighting. Meanwhile, it is also of great interest to explore the utilization of spurious attributes as a means to enhance model performance rather than as impediments to be overcome.

This paper thus aims to develop a novel method of learning models for better defending the SC-GI type of subpopulation shifts from the aspect of innovating the mechanism of dealing with spurious attributes. Considering the object recognition task with backgrounds, addressing SC can be translated to a more straightforward question - Can we design a model capable of reliably recognizing the same object across a range of spurious attributes, such as diverse backgrounds? To answer this question implies that the developed model needs to focus on the invariant and causal features pertinent to object identification, as conceptually illustrated in Figures 1a~1c. Conducting such a study leads to two further pivotal technical questions:

*Q1: How to synthesize samples that vary in spurious attributes yet retain accurate class information?*

*Q2: Based on the synthesized samples, how to design a learning mechanism for developing models more powerful for addressing the SC-GI subpopulation shifts?*

To make a response, a novel framework, the **Correlation Oriented Disentanglement and Augmentation (CODA)**, is developed in this work to develop machine learning models better handling the SC-GI subpopulation shifts by disentangling variant and invariant features, coordinated with a strategic sample augmentation process. The overview of the CODA framework is shown in Figure 2. Two unique developments in CODA are briefed as follows:

**Correlation-Oriented Disentanglement (COD)**  To facilitate the disentangling between class information and spurious information, a novel bi-branch encoding process called Correlation-Oriented Disentanglement (COD) is developed in this paper. As shown in Figures 1d~1f, COD operates through a 'trap and disentangle' mechanism. The variance encoder, coordinated with a decoy classifier that predicts the spurious attribute and a novel design of disentangling loss, extracts only the necessary information about the spurious attributes. The decoder and a reconstruction loss then simultaneously regularize the invariance encoder to capture distinct and complementary information (including invariant features and label-irrelevant features). These bi-branch encoders, coupled with the learned decoder, facilitate the generation of a rich and diverse synthetic dataset.

**Strategic Sample Augmentation with Reweighted Consistency Loss**  Building on insights from recent studies [29, 13], which highlight the effectiveness of upweighting sampling probabilities or losses for minority groups, we extend this concept by strategically generating additional synthetic samples for these groups. By integrating variant and invariant features extracted from different samples, the decoder can generate realistic synthetic data that exhibit different spurious information while preserving class fidelity, aligning with the insight shown in Figures 1a~1c. To complement this, we introduce a novel weighted consistency loss designed to prompt the classifier to deliver uniform predictions across both the original and synthetic samples. Our experimental results affirm that CODA harmonizes effectively with pre-existing robust classification methodologies by integrating the proposed weighted consistency objective.

To the best of our knowledge, the work most akin to ours from the conceptual aspect is CAMEL [9], which applies CycleGAN for learning image transformations. CODA clearly differs from CAMEL in the following aspects: (1) CycleGAN is deterministic and yields a single fixed output for a given input sample, while CODA introduces substantially more flexibility in synthetic data by mixing variant features from multiple samples. (2) CycleGAN is limited to one-to-one transformations, necessitating $\binom{n}{2}$ models to cover all potential mappings for $n$ attribute values within one spurious attribute. In contrast, CODA can simultaneously learn multiple transformations and can be readily extended to accommodate multiple spurious attributes by incorporating additional decoy classifiers. (3) GANs are notoriously challenging to train with instability issues. In contrast, CODA offers a more stable and expedient training process.

The primary contributions of this paper are as follows: (1) The potential of better utilizing spurious attributes, rather than treating them as impediments to be overcome, is explored. A novel correlation-oriented disentanglement and augmentation framework for handling the SC-GI type of subpopulation shifts is thus developed. (2) A bi-branch encoder network architecture coordinated with a decoy classifier and subsequent decoder is designed to realize disentangled representation learning for handling SC. (3) A strategic sample augmentation with a novel weighted consistency loss proposed for upweighting samples predicted with low confidence is developed to better handle the SC-GI. The proposed CODA framework can operate synergistically with existing robust classification methods. (4) Extensive experiments conducted confirm that CODA can drive the learning to focus on causal rather than spurious features and uplift the performance on defending extreme SC-GI subpopulation shifts.

## 2 Preliminaries

### 2.1 Formulation of learning models defending subpopulation shifts

Consider classification tasks in which we aim to predict a label $y \in \mathcal{Y}$ based on an input $x \in \mathcal{X}$. Each instance in the dataset is also characterized by an attribute $a \in \mathcal{A}$, which may influence the prediction. We define a group by a tuple $g = (y, a) \in \mathcal{G}$, where each group has its own distribution $P_g$.

Then the training and testing data distributions can be expressed as mixtures of these subpopulation distributions: $P_{\text{train}} = \sum_{i=1}^{|\mathcal{G}|} \alpha_i P_i$ and $P_{\text{test}} = \sum_{i=1}^{|\mathcal{G}|} \beta_i P_i$, where $\alpha, \beta \in \Delta_{|\mathcal{G}|}$ represent the mixture weights. Subpopulation shifts occur when the mixture weights for training and testing differ, i.e., $\alpha \neq \beta$. If the mixture weights of specific groups dominate, group imbalance occurs, potentially leading to a spurious correlation between the attribute and the label.

Given a loss function $\ell$, our objective is to identify a function $f : \mathcal{X} \to \mathcal{Y}$ that minimizes the worst-case expected loss over all possible mixtures [25], formally defined as:

$$f^* = \underset{f}{\text{argmin}} \sup_{\gamma \in \Delta_{|\mathcal{G}|}} \mathbb{E}_{(x,y) \sim \sum_{i=1}^{|\mathcal{G}|} \gamma_i P_i} [\ell(y, f(x))]. \tag{1}$$

## 2.2 Benchmarks for defending subpopulation shifts

**Empirical Risk Minimization (ERM)**  Consider a training dataset $D = \{(x_i, y_i, a_i)\}_{i=1}^{N}$ comprising $N$ samples. Let $\hat{P}_{\text{train}}$ denote the empirical training distribution. Then common machine learning practice applying ERM aims to find a function $f$ that minimizes the average empirical loss:

$$f^*_{\text{ERM}} = \underset{f}{\text{argmin}} \, \mathbb{E}_{(x,y) \sim \hat{P}_{\text{train}}} [\ell(y, f(x))]. \tag{2}$$

However, ERM typically fails in scenarios with subpopulation shifts due to group imbalance and learning spurious correlations that do not hold in testing data.

**Reweighting Groups (RWG)**  Given the highly imbalanced nature of subpopulation shifts, Idrissi et al. [13] studied the efficacy of data balancing techniques, especially RWG. RWG adopts a group-balanced sampling strategy to upweight minority groups, as shown in Eq. (3):

$$p_g = \frac{1}{|\mathcal{G}| \times N_g}, \tag{3}$$

where $N_g$ is the number of training samples in group $g$, and $p_g$ represents the probability of sampling an instance from group $g$. Despite its simplicity, RWG achieves competitive performance and serves as a strong baseline in studying subpopulation shifts.

**Group Distributionally Robust Optimization (GDRO)**  The study [25] showed that optimizing Eq. (1) is equivalent to minimizing the maximum expected loss over all groups. The empirical optimization objective in GDRO is expressed as:

$$f^*_{\text{GDRO}} = \underset{f}{\text{argmin}} \max_{g \in \mathcal{G}} \mathbb{E}_{(x,y) \sim \hat{P}_g} [\ell(y, f(x))]. \tag{4}$$

## 2.3 Disentangled representation learning

Disentangled representation learning targets extracting data representations that separate the distinct sources of variation and serves as the base of the COD development in this work. This section explores relevant models that have generated significant impacts on such a topic.

**Variational Autoencoder (VAE) and its variant**  Kingma and Welling [18] postulated that the data generative process is governed by an unobserved vector of latent codes $z$ with prior distribution $p(z)$. The Vanilla Variational Autoencoder (VAE) is introduced to approximate the intractable true posterior with a variational posterior $q_\phi(z|x)$ and to learn a decoder $p_\theta(x|z)$, with parameters $\phi$ and $\theta$ respectively. VAE's objective is to maximize:

$$\mathcal{L}_{\text{VAE}}(\theta, \phi) = \mathbb{E}_{p_{\text{data}}(x)} [\mathbb{E}_{q_\phi(z|x)} [\log(p_\theta(x|z)] - \text{KL}(q_\phi(z|x)\|p(z))] \tag{5}$$

$\beta$-VAE [11] incorporated a hyperparameter $\beta$ to the KL term in Eq. (5) to control the trade-off between the reconstruction quality and the independence intensity between latent variables. When $\beta > 1$, the model prioritizes learning independent latent factors.

The study [12] showed that the KL term in Eq. (5) can be decomposed into two components:

$$\mathbb{E}_{p_{\text{data}}(x)} [\text{KL}(q_\phi(z|x)\|p(z))] = I(x; z) + \text{KL}(q_\phi(z)\|p(z)), \tag{6}$$

in which $q_\phi(z) = \mathbb{E}_{p_{\text{data}}(x)}[q_\phi(z|x)]$, and $I(x;z)$ represents the mutual information between $x$ and $z$ under the distribution of $p_{\text{data}}(x)q(z|x)$. A higher value of $\beta$ enforces a closer approximation of the latent codes to the factorized prior $p(z)$, leading to more independent latent codes. However, this also imposes a stronger penalty on the mutual information $I(x;z)$, which may reduce the amount of information about $x$ retained in $z$ and potentially worsen the reconstruction quality.

**Flexibly Fair VAE (FFVAE)** FFVAE [3] aimed to learn feature representations $t$ that achieve subgroup fairness and $z$ targeted for sensitive attributes, with the following objective:

$$\mathcal{L}_{\text{FFVAE}}(\theta, \phi, \psi) = \mathbb{E}_{p_{\text{data}}(x)}[\mathbb{E}_{q_\phi(z,t|x)}[\log p_\theta(x|z,t) + \alpha \log p_\psi(a|z)]$$
$$- \text{KL}(q_\phi(z,t|x)\|p(z,t))] - \gamma \text{KL}(q(z,t)\|q(t)\prod_j q(z_j)), \qquad (7)$$

in which $p_\psi(a|z)$ models the prediction of sensitive attributes. The $\gamma$-weighted KL divergence term encourages the latent variables $z_j$ to be independent of both $t$ and $z_i$ for all $i \neq j$.

Due to the intractability of the second KL term in Eq. (7), FFVAE employs adversarial density ratio estimation [16]. If an estimator cannot differentiate whether a sample pair $(z, t)$ comes from $q(z, t)$ or the factorized distribution $q(t)\prod_j q(z_j)$, the second KL term is minimized, suggesting the independence of $z$ and $t$. However, this approach inevitably introduces a new density estimator component, along with a GAN training objective, which can be challenging and unstable to train.

# 3 Correlation-Oriented Disentanglement and Augmentation (CODA)

This section explains the novel mechanism of CODA on utilizing spurious attributes for defending the SC-GI subpopulation shifts, which is composed of two parts, disentangling variant and invariant features as well as a strategic sample augmentation with a reweighted consistency loss based on the disentangled features.

## 3.1 Correlation-Oriented Disentanglement (COD)

Given a sample $(x, y, a, g)$, we aim to learn encodings $z \in \mathbb{R}^{K_1}$ and $t \in \mathbb{R}^{K_2}$, defined by stochastic parametric encoders $q_\psi(z|x) = \mathcal{N}(z|\mu_z(x), \sigma_z^2(x))$ and $q_\phi(t|x) = \mathcal{N}(t|\mu_t(x), \sigma_t^2(x))$ respectively. Factorial priors $p(z)$ for $z$ and $p(t)$ for $t$ are assumed. Our model includes a decoder, denoted by $p_\theta(x|z,t)$, and a decoy classifier $p_\omega(a|z)$, which predicts the spurious attribute $a$.

In this paper, we propose the following training objective for disentangled representation learning:

$$\mathcal{L}_{\text{COD}}(\theta, \phi, \psi, \omega) = \mathbb{E}_{p_{\text{data}}(x)}[\mathbb{E}_{q_\psi(z|x)q_\phi(t|x)}[\log p_\theta(x|z,t) + \alpha \log p_\omega(a|z)]$$
$$- \gamma \text{KL}(q_\psi(z|x)\|p(z)) - \text{KL}(q_\phi(t|x)\|p(t))]. \qquad (8)$$

By applying Eq. (6), Eq. (8) can be re-written as:

$$\mathcal{L}_{\text{COD}}(\theta, \phi, \psi, \omega) = \mathbb{E}_{p_{\text{data}}(x)}[\mathbb{E}_{q_\psi(z|x)q_\phi(t|x)}[\underbrace{\log p_\theta(x|z,t)}_{\text{reconstruction loss}} + \alpha \underbrace{\log p_\omega(a|z)}_{\text{classification loss}}]]$$
$$- [\gamma \underbrace{I(x;z) + I(x;t)}_{\text{mutual information}}] - [\gamma \underbrace{\text{KL}(q_\psi(z)\|p(z)) + \text{KL}(q_\phi(t)\|p(t))}_{\text{KL divergence losses}}]. \qquad (9)$$

To clarify, Eq. (8) is the practical training objective for implementation, while the equivalent Eq. (9) offers justification for motivation of the proposed COD. The objective function in Eq. (9) consists of four components: (1) a reconstruction loss to ensure $z$ and $t$ collectively encode the information in $x$, (2) a classification loss that encourages $z$ to predict spurious attributes accurately, (3) mutual information terms that limit the extent to which encoders capture relevant information from $x$, and (4) KL divergence terms that penalize the encoders if they diverge from the factorial priors.

The working mechanism of the four components in Eq. (9) is described as follows: The classification loss term rewards $z$ to be maximally predictive of the spurious attribute. By assigning heavy weight ($\gamma > 1$) on minimizing $I(x;z)$, we limit the expressive power of $z$, ensuring it encodes only the necessary information about the spurious attributes. In contrast, we place a relatively low limit on minimizing $I(x;t)$, thus ensuring that the reconstruction quality of the decoder is not degraded. While

the reconstruction loss term ensures that $z$ and $t$ collectively capture all the information contained in $x$, $t$ is expected to carry distinct and complementary information. Furthermore, the two KL terms regularize the variational marginals to match the factorial priors, thus facilitating disentangling within each feature representation. A preliminary theoretical exploration of the performance guarantee of the decoder training has been conducted with a mathematical proof, which is offered in Appendix A.

We emphasize that our goal is to separate spurious and variant correlations from those that are causal and invariant, rather than to seek disentanglement within each individual representation, i.e., imposing each $t_i$ to be independent of $t_j$ (for all $j \neq i$). Consequently, a lower weight on the final KL divergence term is both justified and sufficient, as it supports our objective without imposing unnecessary constraints on the independence within the invariant feature space. Our proposed method circumvents the need for density estimation, thus avoiding the complexities and instabilities associated with training objectives based on the GAN-like density estimator.

### 3.2 Strategic sample augmentation with reweighted consistency loss

**Strategic sample augmentation**  Through the disentanglement of variant and invariant features, as outlined in the previous section, we can now strategically generate synthetic samples for augmentation by leveraging the trained decoder. For a given batch of $B$ samples $\{(x_i, y_i, a_i, g_i)\}_{i=1}^B$, we begin by extracting feature representation pairs $\{(z_i, t_i)\}_{i=1}^B$, where $z_i = q_\psi(z|x_i)$ and $t_i = q_\phi(t|x_i)$. With hyperparameters $L$, which controls the number of synthetic samples per batch instance, we randomly select $L$ indices from $\{1, ..., B\}$ for each instance, yielding mixing set $\{\{h_{i,j}\}_{j=1}^L\}_{i=1}^B$. By combining features from different instances, we create the set of synthetic samples $\{\{\hat{x}_{i,j}\}_{j=1}^L\}_{i=1}^B$, where $\hat{x}_{i,j} = p_\theta(x|z_{h_{i,j}}, t_i)$. These samples exhibit varied spurious information but retain their original class information. A model trained to perform well across augmented samples is likely to learn robust decision-making rules that do not depend on spurious correlations.

**Reweighted consistency loss**  A natural intuition to prevent the final model $f_\xi$ from utilizing spurious correlations is to enforce the model to deliver consistent predictions on the original samples $x_i$ and the set of synthesized samples $\hat{x}_i = \{\hat{x}_{i,j}\}_{j=1}^L$. To realize, we propose the following reweighted consistency loss:

$$\ell_{\text{RC}}(x_i, \hat{x}_i, y_i) = \frac{1}{2}\left[\frac{1}{L}\sum_{i=1}^L (1 - f_\xi(\hat{x}_{i,j})_{y_i})^\beta \text{KL}(f_\xi(\hat{x}_{i,j})\|\tilde{m}_i) + \text{KL}(f_\xi(x_i)\|\tilde{m}_i)\right], \qquad (10)$$

in which $f_\xi(\hat{x}_{i,j})_{y_i}$ is the predicted probability of the true class for $\hat{x}_{i,j}$, and $\tilde{m}_i = Normalize(\frac{1}{L}\sum_{j=1}^L f_\xi(\hat{x}_{i,j})_{y_i}^\beta f_\xi(\hat{x}_{i,j}))$ is the weighted mean prediction for the synthesized samples, normalized to ensure that the probabilities sum to one.

In Eq. (10), we downweight the synthesized samples with low confidence in predicting the right class to mitigate its effect in jeopardizing $\tilde{m}_i$. Conversely, we upweight their KL losses to penalize inconsistencies with other samples that have greater confidence in the ground-truth class.

At last, given loss function $\ell$, we formulate our final optimization objective as follows:

$$\ell_{CODA}(x_i, \hat{x}_i, y_i) = \ell(y_i, f_\xi(x_i)) + \lambda \ell_{\text{RC}}(x_i, \hat{x}_i, y_i). \qquad (11)$$

Pseudo codes of training and deployment of the proposed CODA are offered in Appendix B.

## 4 Experiments

In this section, we evaluate the efficacy of the proposed CODA methodology. Specifically, we aim to answer two questions: (1) Can CODA learn disentangled encodings that extract spurious and causal correlations, respectively? (Section 4.2); (2) Does CODA enhance robustness in addressing SC-GI subpopulation shifts, and what are the critical factors contributing to its effectiveness? (Section 4.3)

### 4.1 Setup

**Dataset description**  Experiments are conducted on the ColoredMNIST and CelebA datasets [22]. Here we provide a brief introduction, with comprehensive details available in Table 1 and Appendix C:

Table 1: Dataset statistics. $SC$ indicates spurious correlations, $GI$ indicates group imbalance, and $DGPS$ indicates the degree of group proportion shifts between training and testing sets.

| | | Group 1 | Group 2 | Group 3 | Group 4 | $SC$ | $GI$ | $DGPS$ |
|---|---|---|---|---|---|---|---|---|
| ColoredMNIST | Training | 21330 (42.26%) | 3776 (7.55%) | 3724 (7.45%) | 21370 (42.74%) | | | |
| | Validation | 2510 (25.1%) | 2526 (25.3%) | 2474 (24.7%) | 2490 (24.9%) | ✓ | ✓ | High |
| | Testing | 761 (7.61%) | 4378 (43.8%) | 4122 (41.2%) | 739 (7.39%) | | | |
| CelebA | Training | 71629 (44.01%) | 22880 (14.06%) | 66874 (41.08%) | 1387 (0.85%) | | | |
| | Validation | 8535 (42.96%) | 2874 (14.47%) | 8276 (41.66%) | 182 (0.92%) | ✓ | ✓ | Low |
| | Testing | 9767 (48.93%) | 2480 (12.42%) | 7535 (37.75%) | 180 (0.90%) | | | |

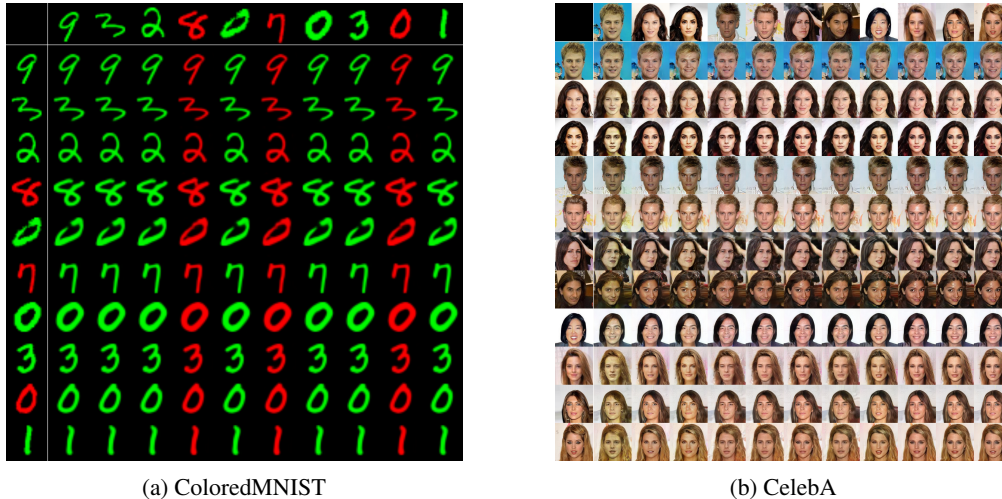

(a) ColoredMNIST            (b) CelebA

Figure 3: Visualization of the synthesized samples. Images from the top-row and the leftmost column are real samples, while the remaining images are reconstructions. Each reconstructed image is generated by combining $t$ extracted from the corresponding leftmost sample and $z$ from the corresponding top-row sample. The main diagonal images represent same sample reconstructions.

- **ColoredMNIST**: This variant of the MNIST dataset [6] incorporates color (red or green) into the original grayscale digits. Digits below five are labeled negative, while the rest are positive. Within the training set, 85% of the negative/positive samples are painted green/red, introducing color as a spurious correlation. Conversely, in the testing set, this ratio is reduced to 15%. Following [1], labels are flipped with a probability of 0.25.
- **CelebA**: The CelebA dataset consists of a large collection of celebrity face images coupled with 40 binary attributes. The task is to predict hair color (blond or non-blond), which exhibits a strong spurious correlation with the gender attribute (male or female).

**Evaluation protocol**    In alignment with [25], we evaluate robustness of all methods using worst-group accuracy, the lowest test accuracy across all groups, as gold-standard. In addition, we also report the average accuracy and the maximum group accuracy gap. Model selection is based on the highest worst-group accuracy obtained during validation.

**Hyperparameters tuning**    For COD, we employ Adam optimizer [17] and adopt pre-trained ResNet-18 and ResNet-50 encoders for ColoredMNIST and CelebA, respectively. For learning the robust classifier, we train all approaches on both datasets using SGD optimizer [24] with momentum 0.9, and adopt a 3-layer CNN and a pre-trained ResNet-50 for ColoredMNIST and CelebA on all approaches, respectively. Additional training details are available in Appendix D.

### 4.2 Disentangling and synthesis abilities of CODA

**CODA can generate samples varying in spurious attributes yet retaining accurate class information.**    Generated samples visualized in Figure 3 well justify the superior disentangling and synthesis capabilities of CODA. It is clear that the generated samples in Figure 3 are visually appealing. It is

Table 2: Average test accuracy (%), worst-group test accuracy (%), and maximum test group accuracy gap (%) of the classifiers on the ColoredMNIST dataset. Variance and invariance classifiers take z and t as inputs, respectively. Results are averaged over three independent trials.

| Classifier | target = Y (digit $\geq$ 5 or not) | | | target = A (color) | | |
|---|---|---|---|---|---|---|
| | Avg. Acc. | Worst. Acc. | Max. Acc. Gap | Avg. Acc. | Worst. Acc. | Max. Acc. Gap |
| variance classifier | $15.00 \pm 0.00$ | $0.00 \pm 0.00$ | $100.00 \pm 0.00$ | $100.00 \pm 0.00$ | $100.00 \pm 0.00$ | $0.00 \pm 0.00$ |
| invariance classifier | $72.17 \pm 0.31$ | $71.29 \pm 0.44$ | $2.64 \pm 0.83$ | $41.44 \pm 0.21$ | $35.73 \pm 0.35$ | $35.60 \pm 1.24$ |

Table 3: Performance metrics (%) for ColoredMNIST and CelebA. Results are averaged over three independent trials. Red/blue/green represents the first/second/third highest performance. ↑ indicates that CODA+X shows improvements over a base method denoted by X in terms of worst-group accuracy and maximum group accuracy on both datasets, X $\in$ {ERM, RWG, GDRO}.

| method | ColoredMNIST | | | CelebA | | |
|---|---|---|---|---|---|---|
| | Avg. Acc. | Worst. Acc. | Max. Acc. Gap | Avg. Acc. | Worst. Acc. | Max. Acc. Gap |
| LfF [23] | $67.64 \pm 5.19$ | $50.91 \pm 2.92$ | $24.36 \pm 7.08$ | $89.67 \pm 0.44$ | $73.11 \pm 1.43$ | $20.72 \pm 2.40$ |
| JTT [21] | $72.11 \pm 0.36$ | $71.01 \pm 0.50$ | $5.15 \pm 1.83$ | $92.10 \pm 0.26$ | $76.45 \pm 0.75$ | $20.90 \pm 0.92$ |
| ERM | $17.02 \pm 0.76$ | $2.08 \pm 0.70$ | $92.70 \pm 0.71$ | $95.05 \pm 0.38$ | $47.78 \pm 2.30$ | $51.80 \pm 2.45$ |
| RWG [13] | $72.94 \pm 0.45$ | $71.64 \pm 0.15$ | $2.70 \pm 1.06$ | $92.87 \pm 0.23$ | $83.44 \pm 1.55$ | $10.78 \pm 1.70$ |
| GDRO [25] | $73.03 \pm 0.25$ | $71.08 \pm 1.02$ | $3.27 \pm 1.13$ | $93.01 \pm 0.08$ | $88.22 \pm 0.89$ | $4.92 \pm 1.11$ |
| CODA+ERM ↑ | $72.20 \pm 0.57$ | $71.74 \pm 0.24$ | $2.45 \pm 1.02$ | $91.89 \pm 0.35$ | $83.65 \pm 0.32$ | $9.87 \pm 1.08$ |
| CODA+RWG ↑ | $73.20 \pm 0.12$ | $72.11 \pm 0.51$ | $2.36 \pm 0.67$ | $91.72 \pm 0.12$ | $86.56 \pm 0.82$ | $7.28 \pm 1.01$ |
| CODA+GDRO ↑ | $73.02 \pm 0.23$ | $71.98 \pm 0.57$ | $2.37 \pm 0.94$ | $90.91 \pm 0.24$ | $89.26 \pm 0.26$ | $4.01 \pm 0.24$ |

Table 4: Performance metrics (%) for ColoredMNIST v2, v3, and v4.

| method | ColoredMNIST v2 | | ColoredMNIST v3 | | ColoredMNIST v4 | |
|---|---|---|---|---|---|---|
| | Worst. Acc. | Max. Acc. Gap | Worst. Acc. | Max. Acc. Gap | Worst. Acc. | Max. Acc. Gap |
| LFF | $58.99 \pm 1.95$ | $12.83 \pm 4.38$ | $54.68 \pm 7.32$ | $15.32 \pm 10.65$ | $49.38 \pm 4.23$ | $21.35 \pm 9.29$ |
| JTT | $71.09 \pm 0.73$ | $2.59 \pm 1.83$ | $69.81 \pm 0.79$ | $6.9 \pm 0.63$ | $62.07 \pm 0.52$ | $12.82 \pm 1.84$ |
| ERM | $9.63 \pm 2.94$ | $87.07 \pm 3.5$ | $0.02 \pm 0.02$ | $99.98 \pm 0.02$ | $0.00 \pm 0.00$ | $100.00 \pm 0.00$ |
| RWG | $71.33 \pm 0.45$ | $2.71 \pm 0.46$ | $70.93 \pm 1.22$ | $2.8 \pm 1.33$ | $68.18 \pm 2.59$ | $8.08 \pm 1.23$ |
| GDRO | $71.84 \pm 0.65$ | $3.32 \pm 0.27$ | $69.65 \pm 1.59$ | $3.96 \pm 2.05$ | $70.42 \pm 0.78$ | $4.17 \pm 0.75$ |
| CODA+ERM ↑ | $71.31 \pm 0.84$ | $3.3 \pm 0.58$ | $71.12 \pm 0.92$ | $3.21 \pm 1.91$ | $71.03 \pm 0.53$ | $3.59 \pm 0.53$ |
| CODA+RWG ↑ | $71.72 \pm 0.95$ | $2.58 \pm 1.51$ | $71.91 \pm 0.21$ | $2.42 \pm 0.32$ | $70.60 \pm 1.22$ | $2.42 \pm 0.93$ |
| CODA+GDRO ↑ | $72.05 \pm 0.27$ | $1.91 \pm 0.44$ | $71.86 \pm 0.58$ | $2.09 \pm 0.22$ | $71.44 \pm 0.69$ | $2.36 \pm 0.54$ |

evident that the digit color/gender in the reconstructed images is consistent with the top-row samples, while the residual features—such as digit, written style, and size/background, posture, hairstyle, and hair color—are determined by the leftmost samples. This indicates that the variance encoder captures only the necessary information about the spurious attribute, while the invariance encoder encapsulates distinct and complementary information, including class-related and irrelevant details. These findings are in harmony with our training objectives outlined in Section 3.1.

**CODA can separate spurious information from causal information.** We further evaluate the ability of CODA to disentangle spurious and causal information through four classification tasks on ColoredMNIST. In each task, we fix the weights of the learned variance and invariance encoders and extract the corresponding latent codes ($z$ and $t$) for each image. Here $z$ and $t$ are expected to carry color and digit information, respectively. A variance classifier and an invariance classifier, both structured as simple 3-layer MLPs, are then employed with: one using $z$ as input and the other using $t$ to predict the digit label (whether the digit $\geq$ 5) or the spurious attribute (color).

Results are presented in Table 2. We observe that the variance classifier achieved perfect performance in predicting color. In contrast, it achieved 0% worst-group accuracy and 15% average group accuracy in predicting digit, which conformed to the color assignment ratio of 15% in the testing set. Conversely, the invariance classifier exhibited robust label classification capabilities, even without applying any reweighting or sample augmentation technique. Its poor color classification performance, comparable to random guessing, justifies the effective disentanglement achieved by CODA.

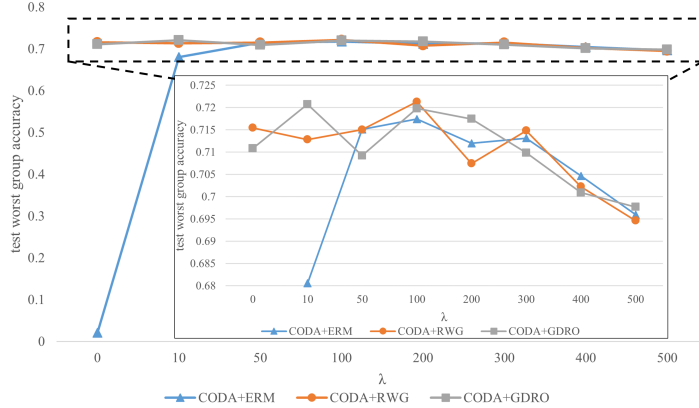

Figure 4: Sensitivity analysis on the weight of the reweighted consistency loss. When $\lambda = 0$, the methods degrade to vanilla ERM, RWG, and GDRO.

Table 5: Sensitivity analysis on the number of synthesized samples per instance on MultipleColoredMNIST. When $L = 0$, the methods degrade to vanilla ERM, RWG, and GDRO.

| Methods | $L$ | Avg. Acc. (std) | Worst Acc. (std) |
|---------|-----|-----------------|------------------|
| CODA+ERM | 0 | $16.60 \pm 1.23$ | $0.00 \pm 0.00$ |
| | 1 | $96.90 \pm 0.29$ | $88.78 \pm 0.87$ |
| | 2 | $97.12 \pm 0.07$ | $90.36 \pm 0.68$ |
| | 4 | $97.06 \pm 0.11$ | $91.85 \pm 0.68$ |
| CODA+RWG | 0 | $95.54 \pm 0.24$ | $85.46 \pm 0.44$ |
| | 1 | $96.92 \pm 0.04$ | $90.87 \pm 0.93$ |
| | 2 | $96.66 \pm 0.28$ | $91.35 \pm 0.70$ |
| | 4 | $96.94 \pm 0.08$ | $91.10 \pm 1.16$ |
| CODA+GDRO | 0 | $94.99 \pm 0.33$ | $82.16 \pm 4.60$ |
| | 1 | $96.68 \pm 0.03$ | $89.88 \pm 0.50$ |
| | 2 | $96.70 \pm 0.16$ | $90.10 \pm 1.10$ |
| | 4 | $96.85 \pm 0.05$ | $90.42 \pm 1.39$ |

## 4.3 Benchmarking studies and analysis

**CODA enhances performance and robustness over existing robust classification methods.** Table 3 illustrates the comprehensive performance metrics for CODA compared with benchmarking methods. On ColoredMNIST, ERM performed worst in all metrics due to the intense subpopulation shift and the spurious correlation. While on CelebA, which has a less intense difference in group proportions, ERM achieved the highest average accuracy but failed in worst-group accuracy due to group imbalance and the spurious correlation. We observe that CODA consistently offered improved worst-group accuracy over vanilla ERM, RWG, and GDRO, and significantly curtailed the group accuracy discrepancy across both datasets. This enhancement in robust accuracy suggests that CODA can be effectively integrated with prevailing robust classification methodologies. Scalability of CODA to the multi-classification scenario is further evaluated on a dataset called MultipleColoredMNIST, which is detailed in Appendix E.1. Experimental results in Table 7 present that CODA can be scaled up to scenarios with multiple spurious attribute values, demonstrating its great potential in solving complex subpopulation shifts.

**Enhanced robustness of CODA to extreme SC-GI subpopulation shifts.** Additional experiments are conducted to assess the robustness of CODA against varying degrees of subpopulation shifts. Color bias is introduced with probabilities $p$ and $1 - p$ in training and testing sets, respectively. ColoredMNIST v2, v3, and v4 are then created with $p = 0.8, 0.9,$ and $0.95$, respectively. A higher value of $p$ indicates greater differences in group proportions, increased group imbalance, and a more intense spurious correlation. Other dataset settings are consistent with ColoredMNIST. Experimental results are shown in Table 4 (complete results are presented in Appendix E.2). As the bias intensity increased, CODA maintained more consistent performance metrics compared to other methods,

which suggested that CODA is more robust to extreme subpopulation shifts compared with baseline methods and can effectively adapt its learning to focus on causal rather than spurious features.

**The critical role of the optimal coefficient selection for RWC to the success of CODA.** Figure 4 presents a sensitivity analysis of the parameter $\lambda$ from Eq. (11) on ColoredMNIST dataset. We observe that an elevated $\lambda$ value ($\lambda \geq 100$) compromises model performance due to excessive regularization. Conversely, a diminutive $\lambda$ value fails to contribute significantly owing to insufficient regularization intensity. Thus, an optimal $\lambda$ value is instrumental for the robust learning of the model.

**Trade-off between robust classification performance and computational costs.** Another critical hyperparameter in training CODA is the number of synthetic samples per instance controlled by $L$. A larger $L$ introduces more variability to synthesized samples, driving the model to "forget" the spurious attributes in decision-making. A sensitivity analysis of $L$ on the MultipleColoredMNIST dataset is provided in Table 5. It is observable that increasing $L$ generally improves the worst-group accuracy. However, a larger $L$ also implies higher computational costs. Furthermore, excess $L$ may not introduce more variability (depending on the complexity of the dataset), e.g., $L$ greater than 10 in MultipleColoredMNIST can result in redundant synthesized samples. Thus, the value of $L$ needs to be selected carefully for best handling the trade-off between classification performance and computational efficiency.

## 5    Conclusion

In this study, we introduced CODA, a novel approach designed to enhance the robustness of machine learning models against the SC-GI subpopulation shifts. Our extensive experiments based on the ColoredMNIST and CelebA datasets provided compelling evidence that CODA could successfully disentangle variant and invariant feature representations and, more importantly, utilize these representations for sample augmentation to significantly improve model robustness.

However, it is important to acknowledge some limitations of CODA, notably its reliance on pre-identified spurious attributes. This dependency requires a potentially costly labeling effort. Additionally, while CODA incurs no additional computational costs at deployment, the training phase is more resource-intensive due to the necessity of data augmentation processes.

Despite these limitations, results of this study offered a promising direction for creating machine learning models that robustly generalize beyond biased training distributions and uphold robustness across diverse subpopulations.

## Acknowledgments and Disclosure of Funding

This work was supported in part by the Shenzhen-Hong Kong-Macau Science & Technology Category C Project with No. SGDX20220530111205037, in part by the Hong Kong RGC General Research Fund Project with No. 11213124, in part by Hong Kong ITC Innovation and Technology Fund Project with No. ITS/034/22MS, in part by Guangdong Provincial Basic and Applied Basic Research - Offshore Wind Power Joint Fund Project under Grant 2022A1515240066, in part by Guangdong Province Technological Project with No. 2023A0505030014, and in part by InnoHK initiative, The Government of the HKSAR, and Laboratory for AI-Powered Financial Technologies.

## Footnotes

[2]We ignore the classification loss for simplicity.

## References

[1] Martin Arjovsky, Léon Bottou, Ishaan Gulrajani, and David Lopez-Paz. Invariant risk minimization, 2020.

[2] Sara Beery, Grant van Horn, and Pietro Perona. Recognition in terra incognita, 2018.

[3] Elliot Creager, David Madras, Jörn-Henrik Jacobsen, Marissa Weis, Kevin Swersky, Toniann Pitassi, and Richard Zemel. Flexibly fair representation learning by disentanglement. In *International conference on machine learning*, pages 1436–1445. PMLR, 2019.

[4] Tal Daniel and Aviv Tamar. Soft-introvae: Analyzing and improving the introspective variational autoencoder. In *Proceedings of the IEEE/CVF Conference on Computer Vision and Pattern Recognition*, pages 4391–4400, 2021.

[5] Tal Daniel and Aviv Tamar. Soft-introvae: Analyzing and improving the introspective variational autoencoder, 2021.

[6] Li Deng. The mnist database of handwritten digit images for machine learning research. *IEEE Signal Processing Magazine*, 29(6):141–142, 2012.

[7] Robert Geirhos, Jörn-Henrik Jacobsen, Claudio Michaelis, Richard Zemel, Wieland Brendel, Matthias Bethge, and Felix A. Wichmann. Shortcut learning in deep neural networks. *Nature Machine Intelligence*, 2(11):665–673, November 2020. ISSN 2522-5839. doi: 10.1038/s42256-020-00257-z. URL http://dx.doi.org/10.1038/s42256-020-00257-z.

[8] Robert Geirhos, Patricia Rubisch, Claudio Michaelis, Matthias Bethge, Felix A. Wichmann, and Wieland Brendel. Imagenet-trained cnns are biased towards texture; increasing shape bias improves accuracy and robustness, 2022.

[9] Karan Goel, Albert Gu, Yixuan Li, and Christopher Ré. Model patching: Closing the subgroup performance gap with data augmentation, 2020.

[10] Tatsunori Hashimoto, Megha Srivastava, Hongseok Namkoong, and Percy Liang. Fairness without demographics in repeated loss minimization. In *International Conference on Machine Learning*, pages 1929–1938. PMLR, 2018.

[11] Irina Higgins, Loic Matthey, Arka Pal, Christopher Burgess, Xavier Glorot, Matthew Botvinick, Shakir Mohamed, and Alexander Lerchner. beta-vae: Learning basic visual concepts with a constrained variational framework. In *International conference on learning representations*, 2016.

[12] Matthew D Hoffman and Matthew J Johnson. Elbo surgery: yet another way to carve up the variational evidence lower bound. In *Workshop in Advances in Approximate Bayesian Inference, NIPS*, volume 1, 2016.

[13] Badr Youbi Idrissi, Martin Arjovsky, Mohammad Pezeshki, and David Lopez-Paz. Simple data balancing achieves competitive worst-group-accuracy, 2022.

[14] Pavel Izmailov, Polina Kirichenko, Nate Gruver, and Andrew Gordon Wilson. On feature learning in the presence of spurious correlations, 2022.

[15] Bingyi Kang, Saining Xie, Marcus Rohrbach, Zhicheng Yan, Albert Gordo, Jiashi Feng, and Yannis Kalantidis. Decoupling representation and classifier for long-tailed recognition, 2020.

[16] Hyunjik Kim and Andriy Mnih. Disentangling by factorising. In *International Conference on Machine Learning*, pages 2649–2658. PMLR, 2018.

[17] Diederik P. Kingma and Jimmy Ba. Adam: A method for stochastic optimization, 2017.

[18] Diederik P Kingma and Max Welling. Auto-encoding variational bayes, 2022.

[19] Pang Wei Koh, Shiori Sagawa, Henrik Marklund, Sang Michael Xie, Marvin Zhang, Akshay Balsubramani, Weihua Hu, Michihiro Yasunaga, Richard Lanas Phillips, Sara Beery, Jure Leskovec, Anshul Kundaje, Emma Pierson, Sergey Levine, Chelsea Finn, and Percy Liang. WILDS: A benchmark of in-the-wild distribution shifts. *CoRR*, abs/2012.07421, 2020. URL https://arxiv.org/abs/2012.07421.

[20] Daniel Levy, Yair Carmon, John C. Duchi, and Aaron Sidford. Large-scale methods for distributionally robust optimization, 2020.

[21] Evan Zheran Liu, Behzad Haghgoo, Annie S. Chen, Aditi Raghunathan, Pang Wei Koh, Shiori Sagawa, Percy Liang, and Chelsea Finn. Just train twice: Improving group robustness without training group information, 2021.

[22] Ziwei Liu, Ping Luo, Xiaogang Wang, and Xiaoou Tang. Deep learning face attributes in the wild. In *Proceedings of International Conference on Computer Vision (ICCV)*, December 2015.

[23] Junhyun Nam, Hyuntak Cha, Sungsoo Ahn, Jaeho Lee, and Jinwoo Shin. Learning from failure: Training debiased classifier from biased classifier, 2020.

[24] Sebastian Ruder. An overview of gradient descent optimization algorithms. *arXiv preprint arXiv:1609.04747*, 2016.

[25] Shiori Sagawa, Pang Wei Koh, Tatsunori B. Hashimoto, and Percy Liang. Distributionally robust neural networks for group shifts: On the importance of regularization for worst-case generalization, 2020.

[26] Amrith Setlur, Don Dennis, Benjamin Eysenbach, Aditi Raghunathan, Chelsea Finn, Virginia Smith, and Sergey Levine. Bitrate-constrained dro: Beyond worst case robustness to unknown group shifts, 2023.

[27] Laleh Seyyed-Kalantari, Haoran Zhang, Matthew BA McDermott, Irene Y Chen, and Marzyeh Ghassemi. Underdiagnosis bias of artificial intelligence algorithms applied to chest radiographs in under-served patient populations. *Nature medicine*, 27(12):2176–2182, 2021.

[28] Kai Xiao, Logan Engstrom, Andrew Ilyas, and Aleksander Madry. Noise or signal: The role of image backgrounds in object recognition, 2020.

[29] Yuzhe Yang, Haoran Zhang, Dina Katabi, and Marzyeh Ghassemi. Change is hard: A closer look at subpopulation shift, 2023.

[30] Huaxiu Yao, Yu Wang, Sai Li, Linjun Zhang, Weixin Liang, James Zou, and Chelsea Finn. Improving out-of-distribution robustness via selective augmentation, 2022.

# A Theoretical exploration

In this section, we provide a theoretical exploration of the reconstruction performance property of CODA, which is stated in Lemma 1. Lemma 1 shows that the decoder in CODA converges to an entropy-regularized version of the underlying data distribution.

In the overall model development, the proposed CODA facilitates the robust training of the final classification model by conducting sample augmentation through a VAE-style generative process. VAE-style generative models are known to have a stable training procedure, but they tend to generate blurry images that lack details. To facilitate realistic image generation based on datasets containing samples of more complicated contents (e.g., CelebA), we adopt the Soft-IntroVAE [4] training technique in which the encoder and decoder models are jointly trained in an introspective way. No extra network component like a discriminator is needed in Soft-IntroVAE.

Recall that we aim to learn a factorized variational posterior $q = q(z, t|x) = q_\psi(z|x)q_\phi(t|x)$, and a decoder $d = p_\theta(x|z, t)$. A factorized prior $p(z, t) = p(z)p(t)$ is assumed for the latent codes. Let $p_{data}$ represent the underlying data distribution. We define $p_d(x) = \mathbb{E}p(z, t)[p\theta(x|z, t)]$ as the distribution corresponding to the samples generated by the decoder.

In the proposed CODA, the ELBO [2] is formulated as:

$$ELBO(x; q, d) = \log p_\theta(x|z, t) - \gamma\text{KL}(q_\psi(z|x)\|p(z)) - \text{KL}(q_\phi(t|x)\|p(t)) \tag{12}$$

By applying Soft-IntroVAE, the overall objectives of the variational posterior and the decoder are formulated in Eq. (13) and Eq. (14), respectively:

$$\mathcal{L}_q(q, d) = \mathbb{E}_{p_{\text{data}}(x)}[ELBO(x; q, d)] - \mathbb{E}_{p_d}[\frac{1}{\kappa}\exp(\kappa ELBO(x; q, d))], \tag{13}$$

$$\mathcal{L}_d(q, d) = \mathbb{E}_{p_{\text{data}}(x)}[ELBO(x; q, d)] + \eta\mathbb{E}_{p_d}[ELBO(x; q, d)], \tag{14}$$

in which $\eta \geq 0$ and $\kappa \geq 1$ are hyperparameters for Soft-IntroVAE training.

A Nash equilibrium point $(q^*, d^*)$ satisfies $\mathcal{L}_q(q^*, d^*) \geq \mathcal{L}_q(q, d^*)$ and $\mathcal{L}_d(q^*, d^*) \geq \mathcal{L}_d(q^*, d)$ for all $q$ and $d$. Define $d^*$ as follows:

$$d^* \in \underset{d}{\text{argmin}}\{\text{KL}(p_{\text{data}}(x)\|p_d(x)) + \eta\text{H}(p_d(x))\}, \tag{15}$$

in which $\text{H}(\cdot)$ denote the Shannon entropy.

**Lemma 1.** *Let $d^*$ be defined in Eq. (15). Denote $q^* = p_{d^*}(z, t|x)$. Assume the encoders and the decoder have infinite capacities. Assume that there exists no region where $p_{data}(x) = 0$, and $p_{d^*}(x) \leq p_{data}(x)^{\frac{1}{\kappa+1}}$ for all $x$. Then $(q^*, d^*)$ forms a Nash equilibrium point of the system formulated in Eq. (13) and Eq. (14).*

*Proof.* Considering the factorized property of the variational posterior and the prior, one can easily obtain the following:

$$\text{KL}(q(z, t|x)\|p(z, t)) = \text{KL}(q_\psi(z|x)\|p(z)) + \text{KL}(q_\phi(t|x)\|p(t)). \tag{16}$$

Based on [4], by applying Eq. (16), Eq. (12) can be reformulated as:

$$ELBO(x; q, d) = \log p_d(x) - \text{KL}(q(z, t|x)\|p_d(z, t|x)) - (\gamma - 1)\text{KL}(q_\psi(z|x)\|p(z)) \tag{17}$$

By plugging Eq. (17) into Eq. (13), we have:

$$\mathcal{L}_q(q, d) = \mathbb{E}_{p_{\text{data}}(x)}[\log p_d(x) - \text{KL}(q(z, t|x)\|p_d(z, t|x)) - (\gamma - 1)\text{KL}(q_\psi(z|x)\|p(z))]$$
$$- \mathbb{E}_{p_d}[\frac{1}{\kappa}\exp(\kappa[\log p_d(x) - \text{KL}(q(z, t|x)\|p_d(z, t|x)) - (\gamma - 1)\text{KL}(q_\psi(z|x)\|p(z))])]. \tag{18}$$

Given fixed d, define $q^*(d)$ such that $\mathcal{L}_q(q^*(d), d) \geq \mathcal{L}_q(q, d)$ holds for all q, i.e., $q^*(d)$ is the maximizer of Eq. (18). Now consider the function $g(y) = ay - \frac{b}{\kappa}\exp(\kappa y)$, where $g'(y) = a - b\exp(\kappa y)$, $g''(y) = -\kappa b\exp(\kappa y)$, $a = p_{\text{data}}(x) \geq 0$, $b = p_d^{\kappa+1}(x) \geq 0$, $y = -\text{KL}(q(z, t|x)\|p_d(z, t|x)) - (\gamma - 1)\text{KL}(q_\psi(z|x)\|p(z)) \leq 0$, and hyperparamter $\kappa \geq 1$. We have the following observations:

**Algorithm 1** Correlation-Oriented Disentanglement

---

**Input**: training dataset $D = \{(x_i, y_i, a_i)\}_{i=1}^N$, batch size $B$, encoders $q_\psi(z|x)$ and $q_\phi(t|x)$, decoder $p_\theta(x|z,t)$, decoy classifier $p_\omega(a|z)$, variant encoder KL weight $\gamma$, classification weight $\alpha$.
**Output**: $q_\psi(z|x)$, $q_\phi(t|x)$, and $p_\theta(x|z,t)$.

1: Randomly initialize $\psi$, $\phi$, $\theta$, and $\omega$.
2: **while** stopping criterion not met **do**
3:     Randomly select batch $\{(x_i, y_i, a_i)\}_{i=1}^B$
4:     **for** $i = 1$ to $B$ **do**
5:       $(\mu_{z_i}, \log \sigma_{z_i}^2) \leftarrow q_\psi(z|x_i)$
6:       $(\mu_{t_i}, \log \sigma_{t_i}^2) \leftarrow q_\phi(t|x_i)$
7:       $z_i \leftarrow \text{Reparameterize}(\mu_{z_i}, \log \sigma_{z_i}^2)$
8:       $t_i \leftarrow \text{Reparameterize}(\mu_{t_i}, \log \sigma_{t_i}^2)$
9:       $\hat{x}_i \leftarrow p_\theta(x|z_i, t_i)$
10:      $\hat{a}_i \leftarrow p_\omega(a|z_i)$
11:     **end for**
12:     $\mathcal{L}_{\text{recon}} \leftarrow \frac{1}{B} \sum_{i=1}^B \ell_{BCE}(x_i, \hat{x}_i)$
13:     $\mathcal{L}_{\text{cls}} \leftarrow \frac{1}{B} \sum_{i=1}^B \ell_{CE}(a_i, \hat{a}_i)$
14:     $\mathcal{L}_{\text{zKL}} \leftarrow \frac{\gamma}{B} \sum_{i=1}^B [-\frac{1}{2} \sum_{j=1}^{\dim(z_i)}(1 + \log \sigma_{z_i,j}^2 - \mu_{z_i,j}^2 - \exp(\log \sigma_{z_i,j}^2))]$
15:     $\mathcal{L}_{\text{tKL}} \leftarrow \frac{1}{B} \sum_{i=1}^B [-\frac{1}{2} \sum_{j=1}^{\dim(t_i)}(1 + \log \sigma_{t_i,j}^2 - \mu_{t_i,j}^2 - \exp(\log \sigma_{t_i,j}^2))]$
16:     $\mathcal{L}_{\text{total}} = \mathcal{L}_{\text{recon}} + \alpha\mathcal{L}_{\text{cls}} + \gamma\mathcal{L}_{\text{zKL}} + \mathcal{L}_{\text{tKL}}$
17:     $\psi, \phi, \theta, \omega \leftarrow \text{backpropagate}(\mathcal{L}_{\text{total}})$
18: **end while**
19: **return** $q_\psi(z|x)$, $q_\phi(t|x)$, and $p_\theta(x|z,t)$.

---

- If $b = 0$ and $a > 0$, we have $g'(y) = a \geq 0$, thus the maximum of $g(y)$ is obtained at $y = 0$.

- If $b = 0$ and $a = 0$, we have $g'(y) = 0$, thus $y = 0$ yields a maximizer of $g(y)$.

- If $b > 0$ and $a = 0$, we have $g'(y) < 0$, thus there is no maximizer of $g(y)$.

- If $b > 0$ and $a > 0$, we have saddle point $y^* = \frac{1}{\kappa} \log \frac{a}{b}$. If $\frac{a}{b} \in (0, 1)$, we have $y^* < 0$, thus the maximum of $g(y)$ is obtained at $y^*$. If $\frac{a}{b} \geq 1$, we have $y^* \geq 0$, thus the maximum of $g(y)$ is obtained at 0.

We have $y = 0$ if and only if $q(z, t|x) = p_d(z, t|x)$ and $q_\phi(z|x) = p(z)$. Thus, given the assumption in Lemma 1, for any fixed $d$, $q^*(d)$ satisfies $q^*(d)(z, t|x) = p_d(z, t|x)$ and $q_\psi^*(z|x) = p(z)$. Thus we have $ELBO(x; q^*(d), d) = \log p_d(x)$.

Next, we consider

$$\begin{aligned}
\mathcal{L}_d(q^*(d), d) &= \mathbb{E}_{p_{\text{data}}(x)}[ELBO(x; q^*(d), d)] + \eta\mathbb{E}_{p_d}[ELBO(x; q^*(d), d)] \\
&= \mathbb{E}_{p_{\text{data}}(x)}[\log p_d(x)] + \eta\mathbb{E}_{p_d}[\log p_d(x)] \\
&= -\text{KL}(p_{\text{data}}\|p_d) + \mathbb{E}_{p_{\text{data}}}[\log p_{\text{data}}(x)] - \eta\text{H}(p_d(x)).
\end{aligned} \tag{19}$$

Since $\mathbb{E}_{p_{\text{data}}}[\log p_{\text{data}}(x)]$ is a constant term, we conclude that $d^* \in \text{argmin}_d\{\text{KL}(p_{\text{data}}(x)\|p_d(x)) + \eta\text{H}(p_d(x))\}$, and $(q^*, d^*)$ forms a Nash equilibrium point of the system formulated in Eq. (13) and Eq. (14).

$\square$

# B Pseudo codes

This section presents the pseudo-codes for training the proposed CODA framework. The training process of the CODA framework is comprised of two stages:

- The pseudo-code for training the correlation-oriented disentanglement is detailed in Algorithm 1.

---

**Algorithm 2** Sample Augmentation

---

**Input**: batch size $B$, number of synthetic samples per instance $L$, original batch samples $\{x_i\}_{i=1}^B$, encoders $q_\psi(z|x)$ and $q_\phi(t|x)$, decoder $p_\theta(x|z,t)$.
**Output**: synthesized samples $\{\{\hat{x}_{i,j}\}_{j=1}^L\}_{i=1}^B$.

1: **for** $i = 1$ to $B$ **do**
2:   $(\mu_{z_i}, \log \sigma_{z_i}^2) \leftarrow q_\psi(z|x_i)$
3:   $(\mu_{t_i}, \log \sigma_{t_i}^2) \leftarrow q_\phi(t|x_i)$
4:   $z_i \leftarrow \text{Reparameterize}(\mu_{z_i}, \log \sigma_{z_i}^2)$
5:   $t_i \leftarrow \text{Reparameterize}(\mu_{t_i}, \log \sigma_{t_i}^2)$
6: **end for**
7: **for** $i = 1$ to $B$ **do**
8:   **for** $j = 1$ to $L$ **do**
9:     randomly sample $h$ from $\{1, ..., B\}$
10:     $\hat{x}_{i,j} \leftarrow p_\theta(x|z_h, t_i)$
11:   **end for**
12: **end for**
13: **return** $\{\{\hat{x}_{i,j}\}_{j=1}^L\}_{i=1}^B$

---

---

**Algorithm 3** Robust Classifier with Reweighted Consistency

---

**Input**: training dataset $D = \{(x_i, y_i, a_i)\}_{i=1}^N$, batch size $B$, number of synthetic samples per instance $L$, consistency coefficient $\lambda$, reweight coefficient $\beta$, encoders $q_\psi(z|x)$ and $q_\phi(t|x)$, decoder $p_\theta(x|z,t)$, final classifier $f_\xi$.
**Output**: robust classifier $f_\xi$.

1: Randomly initialize $\xi$.
2: **while** stopping criterion not met **do**
3:   Randomly select batch $\{(x_i, y_i, a_i)\}_{i=1}^B$.
4:   Synthesize samples $\{\{\hat{x}_{i,j}\}_{j=1}^L\}_{i=1}^B$ from Algorithm 2.
5:   **for** $i = 1$ to $B$ **do**
6:     $\hat{y}_i \leftarrow f_\xi(x_i)$
7:     $\ell_{RC,i} \leftarrow \ell_{RC}(x_i, \{\hat{x}_{i,j}\}_{j=1}^L, y_i)$ based on Eq. (10).
8:   **end for**
9:   $\mathcal{L}_{\text{cls}} \leftarrow \frac{1}{B} \sum_{i=1}^B \ell_{CE}(y_i, \hat{y}_i)$
10:   $\mathcal{L}_{\text{consistency}} \leftarrow \frac{1}{B} \sum_{i=1}^B \ell_{RC,i}$
11:   $\mathcal{L}_{\text{total}} \leftarrow \mathcal{L}_{\text{cls}} + \lambda \mathcal{L}_{\text{consistency}}$
12:   $\xi \leftarrow \text{backpropagate}(\mathcal{L}_{\text{total}})$
13: **end while**
14: **return** robust classifier $f_\xi$.

---

- The pseudo-code for training the robust classifier with the proposed reweighted consistency loss in Eq. (10) is outlined in Algorithm 3.

## C   Dataset details

Experiments are conducted on the ColoredMNIST and CelebA datasets. Let $DGPS$ denote the degree of group proportion shifts between training and testing sets. Denote $p_g^{Tr}$ and $p_g^{Te}$ as the proportions of group $g$ in training and testing sets, respectively. We then define the average group proportion shift ($AGPS$) as $\frac{1}{|\mathcal{G}|} \sum_{g \in |\mathcal{G}|} |p_g^{Tr} - p_g^{Te}|$. In this paper, for $|\mathcal{G}| = 4$, we categorize $DGPS$ as high if $AGPS$ exceeds 25%, low if $AGPS$ is below 5%, and medium else. Key statistics of both datasets are summarized in Table 1.

### C.1   ColoredMNIST

The ColoredMNIST dataset is a variant of the classic MNIST dataset of handwritten digits. In ColoredMNIST, samples with a digit less than 5 are labeled as negative samples ($Y = 0$), and those

with a digit 5 or higher are labeled as positive samples ($Y = 1$). Additionally, each sample is colored either red ($A = 1$) or green ($A = 0$). To introduce a spurious correlation, 85% of the positive and negative samples are colored red and green, respectively, in the training set. This bias is adjusted in the validation set to 50% for a balanced distribution and reversed in the test set to 15% to challenge the learned correlation.

The training and validation sets are derived from the first 50,000 and last 10,000 samples of the original MNIST dataset, respectively. An independent test set of 10,000 samples is also provided. Consistent with the work of [1], label noise is added to the dataset, with a 25% flip rate.

Characteristically, as shown in Table 1, ColoredMNIST presents challenges, such as group imbalance, significant variation in group proportions between training and testing sets, and strong spurious correlations between color and label. These features make it an ideal benchmark for testing the robustness and generalization of machine learning models against subpopulation shifts.

## C.2 CelebA

The CelebA dataset is comprised of an extensive collection of celebrity face images, each tagged with 40 binary attributes. The task is to classify the hair color (blond or not-blond), which is highly correlated with gender (male or female). We adhere to the standard splits for training (162,770 samples), validation (19,867 samples), and testing (19,962 samples). Like ColoredMNIST, CelebA also exhibits extreme group imbalance and spurious correlations, though with less variation in group proportions between the training and testing sets compared to ColoredMNIST. These characteristics render CelebA a formidable dataset for assessing the robustness and generalization capabilities of machine learning algorithms.

## D Training details

This section outlines the training parameters and computational details for our experiments. The experiments were conducted on a single NVIDIA L40 GPU.

### D.1 Disentangled representation learning in CODA

For the COD training, we utilized the Adam optimizer [17] and used a consistent batch size of 128 across 50 epochs on both datasets. Pre-trained ResNet-18 and ResNet-50 encoders were employed for the ColoredMNIST and CelebA datasets, respectively. The learning rates were set to $1e - 3$ with a corresponding weight decay of $1e - 3$ for the decoy classifier on both datasets. The learning rates for other network components were $1e - 4$ for ColoredMNIST and $2e - 5$ for CelebA, both with zero weight decay. To enhance visual quality in the generated images on CelebA, we applied the Soft-IntroVAE training method [5] with $\beta_{neg} = 1024$ during the final 20 epochs (batch size is reduced to 48). Parameters $\alpha = 0.1$ and $\gamma = 5$ were used in Eq. (9) for both datasets. The training times were approximately 1 hour and 38 minutes with 14 GB of GPU memory for ColoredMNIST, and 45 hours and 29.4 minutes consuming 45.13 GB of GPU memory for CelebA, respectively.

### D.2 Robust classifier training details

For the robust classifier, we trained all methods using the SGD optimizer [24] with a momentum of 0.9 and used a consistent batch size of 64 across 50 epochs on both datasets. A 3-layer CNN and a pre-trained ResNet-50 were adopted for ColoredMNIST and CelebA, respectively. The learning rate was fixed at $1e - 5$ with a weight decay of $1e - 1$ for CelebA, and both set to $1e - 3$ for ColoredMNIST. The number of augmented samples, denoted by $L$, was set to 2 for ColoredMNIST and 4 for CelebA. Training times on the ColoredMNIST dataset were as follows: CODA+ERM completed in 26.36 minutes, CODA+RWG in 26.28 minutes, and CODA+GDRO in 27.32 minutes, all with a GPU memory consumption of 488 MB. On the CelebA dataset, the training durations were 11.30 hours for CODA+ERM, 11.46 hours for CODA+RWG, and 19.35 hours for CODA+GDRO, each utilizing 36.35 GB of GPU memory. We conducted hyperparameter tuning for various robust classification approaches through a grid search method:

- **LfF:** We grid searched over $q \in \{0.1, 0.3, 0.5, 0.7, 0.9\}$.

Table 6: Average accuracy (%) for ColoredMNIST v2, v3, and v4.

| method | ColoredMNIST v2 | Avg. Acc. ColoredMNIST v3 | ColoredMNIST v4 |
|---|---|---|---|
| LFF | $65.96 \pm 2.0$ | $66.16 \pm 0.6$ | $55.02 \pm 34.00$ |
| JTT | $72.68 \pm 0.56$ | $72.01 \pm 0.25$ | $71.54 \pm 1.32$ |
| ERM | $45.69 \pm 18.9$ | $10.13 \pm 0.01$ | $5.00 \pm 0.00$ |
| RWG | $72.96 \pm 0.14$ | $72.62 \pm 0.07$ | $72.03 \pm 0.76$ |
| GDRO | $72.92 \pm 0.13$ | $72.58 \pm 0.29$ | $72.22 \pm 0.19$ |
| CODA+ERM | $72.87 \pm 0.21$ | $72.3 \pm 0.14$ | $72.51 \pm 0.06$ |
| CODA+RWG | $73.28 \pm 0.12$ | $73.34 \pm 0.16$ | $72.59 \pm 0.33$ |
| CODA+GDRO | $73.19 \pm 0.11$ | $73.01 \pm 0.26$ | $72.98 \pm 0.23$ |

Table 7: Performance metrics (%) for MultipleColoredMNIST.

| | Avg. Acc. (std) | Worst Acc. (std) |
|---|---|---|
| ERM | $16.60 \pm 1.23$ | $0.00 \pm 0.00$ |
| RWG | $95.54 \pm 0.24$ | $85.46 \pm 0.44$ |
| GDRO | $94.99 \pm 0.33$ | $82.16 \pm 4.60$ |
| CODA+ERM | $97.06 \pm 0.11$ | $91.85 \pm 0.68$ |
| CODA+RWG | $96.94 \pm 0.08$ | $91.10 \pm 1.16$ |
| CODA+GDRO | $96.85 \pm 0.05$ | $90.42 \pm 1.39$ |

- **GDRO:** The group adjustment hyperparameter $C$ was selected from $\{1, 2, 3, 4, 5\}$.

- **JTT:** A grid search was performed for the identification epoch $T$ over $\{1, 2, 5\}$ and for the upsampling factor $\lambda_{up}$ over $\{5, 20, 100\}$.

- **CODA:** The hyperparameter $\lambda$ was searched within $\{10, 50, 100, 200, 300, 400, 500\}$ for ColoredMNIST and $\{10, 50, 100\}$ for CelebA.

# E    Additional experiments and analysis

## E.1    Scalability of CODA to the multi-classification scenario

Scalability of CODA to the multi-classification scenario is evaluated on a dataset called Multiple-ColoredMNIST. In this dataset, the task is to predict the digit $y \in \{0, \dots, 9\}$, where $|Y| = 10$. We define ten RGB colors $a \in \{0, \dots, 9\}$ such that $|A| = 10$, resulting in a total of 100 groups. In the training set, each sample is painted with the color corresponding to its digit $a = y$ with a probability of 85%, and randomly assigned another for the remaining 15%. Labels are flipped with a probability of 25%, similar to the ColoredMNIST dataset. The training set is highly group imbalanced, with colors spuriously correlated with labels. The majority group (in a random run) constitutes 9.44% of the population, while the minority group only 0.13%.

The worst group accuracy of the hypothetical optimal digit classifier f can be as low as 62.07% (only 62.07% of the samples whose digits match the labels due to randomness in label flipping and color assignment). Thus, for validation and test sets, we used a different setup, no label flipping and uniform color assignment. The worst group accuracy of f is 100% for both sets.

Experimental results are presented in Table 7. CODA demonstrates substantially higher performance compared to benchmark methods, indicating its scalability to scenarios with multiple spurious attribute values. These results underscore the effectiveness of CODA in addressing complex subpopulation shifts.

## E.2    Robustness of CODA against different degrees of subpopulation shifts

To further assess the robustness of the CODA framework against different degrees of subpopulation shifts, we conducted a series of additional experiments. These experiments were designed to simulate extreme subpopulation shifts by introducing color bias into the training and testing datasets with probabilities $p$ and $1 - p$, respectively. Specifically, we generated three new versions of the ColoredMNIST dataset—versions 2, 3, and 4—with the probabilities $p$ set at 0.8, 0.9, and 0.95,

Table 8: Performance comparison (%) of different encoders to CODA.

| Encoder Type | Method | Avg. Acc. (std) | Worst Acc. (std) |
|---|---|---|---|
| 3-CNN-layer Encoders | CODA+ERM | 68.89 ± 0.59 | 67.50 ± 1.08 |
| | CODA+RWG | 70.89 ± 0.31 | 69.83 ± 0.38 |
| | CODA+GDRO | 71.36 ± 0.48 | 70.51 ± 0.52 |
| Resnet18 Encoders | CODA+ERM | 72.20 ± 0.57 | 71.74 ± 0.24 |
| | CODA+RWG | 73.20 ± 0.12 | 72.11 ± 0.51 |
| | CODA+GDRO | 73.02 ± 0.23 | 71.98 ± 0.57 |

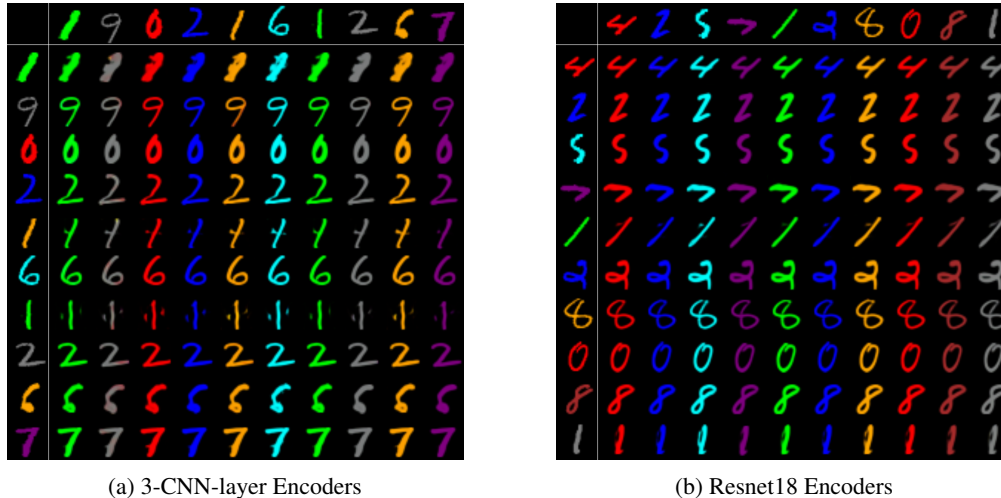

(a) 3-CNN-layer Encoders          (b) Resnet18 Encoders

Figure 5: Visualization of the synthesized samples. Both encoder architectures can well disentangle the causal features from the spurious ones.

respectively. For each version, we maintained a fixed validation ratio of 0.5 to ensure consistency in evaluation across different datasets.

The details of the dataset configurations are fully described in Appendix C.1. The results from these experiments are detailed in Table 4 and Table 6. The findings consistently show that CODA not only enhances the performance of baseline methods but also significantly improves the robustness of the model. Notably, CODA achieved the highest accuracy in the worst-performing groups and the smallest maximum accuracy gap across groups, regardless of the intensity of the subpopulation shifts. These results underline the effectiveness of CODA in handling challenging scenarios where traditional models often falter, further validating its utility in real-world applications with data bias.

### E.3 The critical role of the reconstruction quality to robust classification of CODA

The efficacy of CODA relies on its ability to synthesize samples with varied spurious attribute values alongside the original sample, thereby encouraging the final classifier to focus exclusively on causal features. This section evaluates the impact of the reconstruction quality on robust classification performance.

To facilitate this comparison, a simple 3-CNN-layer architecture was employed for encoders, representing a less sophisticated alternative to the ResNet18 encoders. Evaluation on the Multiple-ColoredMNIST dataset revealed that the 3-CNN-layer architecture produced significantly higher pixel-wise reconstruction losses (0.06451 and 0.05424 for validation and test sets) compared to ResNet18 (0.00055 and 0.00057). Despite this gap in reconstruction quality, both architectures achieved similar disentanglement performance (see Fig 5), successfully separating causal and spurious features.

Table 8 presents the experimental results. The data indicate that the classification performance of CODA using ResNet18 encoders surpasses that of the 3-CNN-layer encoders. This observation suggests a positive correlation between reconstruction quality and final classification performance, with poorer reconstruction quality corresponding to decreased classification accuracy.

